# Overlaying classifiers:
# a practical approach for optimal ranking

**Stéphan Clémençon**
Telecom Paristech (TSI) - LTCI UMR Institut Telecom/CNRS 5141
stephan.clemencon@telecom-paristech.fr

**Nicolas Vayatis**
ENS Cachan & UniverSud - CMLA UMR CNRS 8536
vayatis@cmla.ens-cachan.fr

## Abstract

ROC curves are one of the most widely used displays to evaluate performance of scoring functions. In the paper, we propose a statistical method for directly optimizing the ROC curve. The target is known to be the regression function up to an increasing transformation and this boils down to recovering the level sets of the latter. We propose to use classifiers obtained by empirical risk minimization of a weighted classification error and then to construct a scoring rule by overlaying these classifiers. We show the consistency and rate of convergence to the optimal ROC curve of this procedure in terms of supremum norm and also, as a byproduct of the analysis, we derive an empirical estimate of the optimal ROC curve.

## 1 Introduction

In applications such as medical diagnosis, credit risk screening or information retrieval, one aims at ordering instances under binary label information. The problem of ranking binary classification data is known in the machine learning literature as the bipartite ranking problem ([FISS03], [AGH+05], [CLV08]). A natural approach is to find a real-valued scoring function which mimics the order induced by the regression function. A classical performance measure for scoring functions is the Receiver Operating Characteristic (ROC) curve which plots the rate of true positive against false positive ([vT68], [Ega75]). The ROC curve offers a graphical display which permits to judge rapidly how a scoring rule discriminates the two populations (positive against negative). A scoring rule whose ROC curve is close to the diagonal line does not discriminate at all, while the one lying above all others is the best possible choice. From a statistical learning perspective, risk minimization (or performance maximization) strategies for bipartite ranking have been based mostly on a popular summary of the ROC curve known as the Area Under a ROC Curve (AUC - see [CLV08], [FISS03], [AGH+05]) which corresponds to the $L_1$-metric on the space of ROC curves. In the present paper, we propose a statistical methodology to estimate the optimal ROC curve in a stronger sense than the AUC sense, namely in the sense of the supremum norm. In the same time, we will explain how to build a nearly optimal scoring function. Our approach is based on a simple observation: optimal scoring functions can be represented from the collection of level sets of the regression function. Hence, the bipartite ranking problem may be viewed as a 'continuum' of classification problems with asymmetric costs where the targets are the level sets. In a nonparametric setup, regression or density level sets can be estimated with plug-in methods ([Cav97], [RV06], [AA07], [WN07], ...). Here, we take a different approach based on a weighted empirical risk minimization principle. We provide rates of convergence with which an optimal point of the ROC curve can be recovered according to this principle. We also develop a practical ranking method based on a discretization of the original problem. From the resulting classifiers and their related empirical errors, we show how

to build a linear-by-part estimate of the optimal ROC curve and a quasi-optimal piecewise constant scoring function. Rate bounds in terms of the supremum norm on ROC curves for these procedures are also established.

The rest of the paper is organized as follows: in Section 2, we present the problem and give some properties of ROC curves, in Section 3, we provide a statistical result for the weighted empirical risk minimization, and in Section 4, we develop the main results of the paper which describe the statistical performance of a scoring rule based on overlaying classifiers as well as the rate of convergence of the empirical estimate of the optimal ROC curve.

## 2 Bipartite ranking, scoring rules and ROC curves

**Setup.** We study the ranking problem for classification data with binary labels. The data are assumed to be generated as i.i.d. copies of a random pair $(X, Y) \in \mathcal{X} \times \{-1, +1\}$ where $X$ is a random descriptor living in the measurable space $\mathcal{X}$ and $Y$ represents its binary label (relevant vs. irrelevant, healthy vs. sick, ...). We denote by $P = (\mu, \eta)$ the distribution of $(X, Y)$, where $\mu$ is the marginal distribution of $X$ and $\eta$ is the *regression function* (up to an affine transformation): $\eta(x) = \mathbb{P}\{Y = 1 \mid X = x\}$, $x \in \mathcal{X}$. We will also denote by $p = \mathbb{P}\{Y = 1\}$ the proportion of positive labels. In the sequel, we assume that the distribution $\mu$ is absolutely continuous with respect to Lebesgue measure.

**Optimal scoring rules.** We consider the approach where the ordering can be derived by the means of a *scoring function* $s : \mathcal{X} \to \mathbb{R}$: one expects that the higher the value $s(X)$ is, the more likely the event "$Y = +1$" should be observed. The following definition sets the goal of learning methods in the setup of bipartite ranking.

**Definition 1 (Optimal scoring functions)** *The class of optimal scoring functions is given by the set*

$$\mathcal{S}^* = \{ s^* = T \circ \eta \mid T : [0, 1] \to \mathbb{R} \text{ strictly increasing} \}.$$

Interestingly, it is possible to make the connection between an arbitrary (bounded) optimal scoring function $s^* \in \mathcal{S}^*$ and the distribution $P$ (through the regression function $\eta$) completely explicit.

**Proposition 1 (Optimal scoring functions representation, [CV08])** *A bounded scoring function $s^*$ is optimal if and only if there exist a nonnegative integrable function $w$ and a continuous random variable $V$ in $(0, 1)$ such that:*

$$\forall x \in \mathcal{X}, \quad s^*(x) = \inf_{\mathcal{X}} s^* + \mathbb{E}\left(w(V) \cdot \mathbb{I}\{\eta(x) > V\}\right).$$

A crucial consequence of the last proposition is that solving the bipartite ranking problem amounts to recovering the collection $\{x \in \mathcal{X} \mid \eta(x) > u\}_{u \in (0,1)}$ of level sets of the regression function $\eta$. Hence, the bipartite ranking problem can be seen as a collection of overlaid classification problems. This view was first introduced in [CV07] and the present paper is devoted to the description of a statistical method implementing this idea.

**ROC curves.** We now recall the concept of ROC curve and explain why it is a natural choice of performance measure for the ranking problem with classification data. We consider here only *true* ROC curves which correspond to the situation where the underlying distribution is known. First, we need to introduce some notations. For a given scoring rule $s$, the conditional cdfs of the random variable $s(X)$ are denoted by $G_s$ and $H_s$. We also set, for all $z \in \mathbb{R}$:

$$\bar{G}_s(z) = 1 - G_s(z) = \mathbb{P}\{s(X) > z \mid Y = +1\},$$
$$\bar{H}_s(z) = 1 - H_s(z) = \mathbb{P}\{s(X) > z \mid Y = -1\}.$$

to be the residual conditional cdfs of the random variable $s(X)$. When $s = \eta$, we shall denote the previous functions by $G^*$, $H^*$, $\bar{G}^*$, $\bar{H}^*$ respectively.

We introduce the notation $Q(Z, \alpha)$ to denote the quantile of order $1 - \alpha$ for the distribution of a random variable $Z$ conditioned on the event $Y = -1$. In particular, the following quantile will be of interest:

$$Q^*(\alpha) = Q(\eta(X), \alpha) = \bar{H}^{*-1}(\alpha),$$

where we have used here the notion of generalized inverse $F^{-1}$ of a càdlàg function $F$: $F^{-1}(z) = \inf\{t \in \mathbb{R} \mid F(t) \geq z\}$. We now turn to the definition of the ROC curve.

**Definition 2 (True ROC curve)** *The* ROC *curve of a scoring function $s$ is the parametric curve:*

$$z \mapsto \left(\bar{H}_s(z), \bar{G}_s(z)\right)$$

*for thresholds $z \in \mathbb{R}$. It can also be defined as the plot of the function:*

$$\text{ROC}(s, \cdot) \ : \ \alpha \in [0,1] \mapsto \bar{G}_s \circ \bar{H}_s^{-1}(\alpha) = \bar{G}_s\left(Q(s(X), \alpha)\right) \ .$$

*By convention, points of the curve corresponding to possible jumps (due to possible degenerate points of $H_s$ or $G_s$) are connected by line segments, so that the* ROC *curve is always continuous. For $s = \eta$, we take the notation $\text{ROC}^*(\alpha) = \text{ROC}(\eta, \alpha)$.*

The residual cdf $\bar{G}_s$ is also called the *true positive rate* while $\bar{H}_s$ is the *false positive rate*, so that the ROC curve is the plot of the true positive rate against the false positive rate.

Note that, as a functional criterion, the ROC curve induces a partial order over the space of all scoring functions. Some scoring function might provide a better ranking on some part of the observation space and a worst one on some other. A natural step to take is to consider local properties of the ROC curve in order to focus on best instances but this is not straightforward as explained in [CV07].

We expect optimal scoring functions to be those for which the ROC curve dominates all the others for all $\alpha \in (0,1)$. The next proposition highlights the fact that the ROC curve is relevant when evaluating performance in the bipartite ranking problem.

**Proposition 2** *The class $\mathcal{S}^*$ of optimal scoring functions provides the best possible ranking with respect to the* ROC *curve. Indeed, for any scoring function $s$, we have:*

$$\forall \alpha \in (0,1) \ , \quad \text{ROC}^*(\alpha) \geq \text{ROC}(s, \alpha) \ ,$$

*and $\forall s^* \in \mathcal{S}^*$, $\forall \alpha \in (0,1)$ , $\quad \text{ROC}(s^*, \alpha) = \text{ROC}^*(\alpha)$.*

The following result will be needed later.

**Proposition 3** *We assume that the optimal* ROC *curve is differentiable. Then, we have, for any $\alpha$ such that $Q^*(\alpha) < 1$:*

$$\frac{d}{d\alpha}\text{ROC}^*(\alpha) = \frac{1-p}{p} \cdot \frac{Q^*(\alpha)}{1 - Q^*(\alpha)} \ .$$

For proofs of the previous propositions and more details on true ROC curves, we refer to [CV08].

## 3 Recovering a point on the optimal ROC curve

We consider here the problem of recovering a single point of the optimal ROC curve from a sample of i.i.d. copies $\{(X_i, Y_i)\}_{i=1,\ldots,n}$ of $(X, Y)$. This amounts to recovering a single level set of the regression function $\eta$ but we aim at controlling the error in terms of rates of false positive and true positive. For any measurable set $C \subset \mathcal{X}$, we set the following notations:

$$\alpha(C) = \mathbb{P}(X \in C \mid Y = -1) \ \text{ and } \ \beta(C) = \mathbb{P}(X \in C \mid Y = +1) \ .$$

We also define the weighted classification error:

$$L_\omega(C) = 2p(1 - \omega)\left(1 - \beta(C)\right) + 2(1 - p)\omega\,\alpha(C) \ ,$$

with $\omega \in (0,1)$ being the asymmetry factor.

**Proposition 4** *The optimal set for this error measure is $C_\omega^* = \{x \ : \ \eta(x) > \omega\}$. We have indeed, for all $C \subset \mathcal{X}$:*

$$L_\omega(C_\omega^*) \leq L_\omega(C) \ .$$

*Also the optimal error is given by:*

$$L_\omega(C_\omega^*) = 2\mathbb{E}\min\{\omega(1 - \eta(X)), (1 - \omega)\eta(X)\} \ .$$

*The excess risk for an arbitrary set $C$ can be written:*

$$L_\omega(C) - L_\omega(C_\omega^*) = 2\mathbb{E}\left(\mid \eta(X) - \omega \mid \mathbb{I}\{X \in C\Delta C_\omega^*\}\right) \ ,$$

*where $\Delta$ stands for the symmetric difference between sets.*

The empirical counterpart of the weighted classification error can be defined as:

$$\hat{L}_\omega(C) = \frac{2\omega}{n} \sum_{i=1}^n \mathbb{I}\{Y_i = -1, \ X_i \in C\} + \frac{2(1-\omega)}{n} \sum_{i=1}^n \mathbb{I}\{Y_i = +1, \ X_i \notin C\}\,.$$

This leads to consider the *weighted empirical risk minimizer* over a class $\mathcal{C}$ of candidate sets:

$$\hat{C}_\omega = \arg\min_{C \in \mathcal{C}} \hat{L}_\omega(C).$$

The next result provides rates of of convergence of the weighted empirical risk minimizer $\hat{C}_\omega$ to the best set in the class in terms of the two types of error $\alpha$ and $\beta$.

**Theorem 1** *Let $\omega \in (0,1)$. Assume that $\mathcal{C}$ is of finite VC dimension $V$ and contains $C_\omega^*$. Suppose also that both $G^*$ and $H^*$ are twice continuously differentiable with strictly positive first derivatives and that $\mathrm{ROC}^*$ has a bounded second derivative. Then, for all $\delta > 0$, there exist constants $c(V)$ independent of $\omega$ such that, with probability at least $1 - \delta$:*

$$|\alpha(\hat{C}_\omega) - \alpha(C_\omega^*)| \le \frac{c(V)}{\sqrt{p(1-\omega)}} \cdot \left( \frac{\log(1/\delta)}{n} \right)^{\frac{1}{3}}\,.$$

*The same result also holds for the excess risk of $\hat{C}_\omega$ in terms of the rate $\beta$ of true positive with a factor term of $\sqrt{(1-p)\omega}$ in the denominator instead .*

It is noteworthy that, while convergence in terms of classification error is expected to be of the order of $n^{-1/2}$, its two components corresponding to the rate of false positive and true positive present slower rates.

## 4  Nearly optimal scoring rule based on overlaying classifiers

**Main result.** We now propose to collect the classifiers studied in the previous section in order to build a scoring function for the bipartite ranking problem. From Proposition 1, we can focus on optimal scoring rules of the form:

$$s^*(x) = \int \mathbb{I}\{x \in C_\omega^*\}\, \nu(d\omega), \tag{1}$$

where the integral is taken w.r.t. any positive measure $\nu$ with same support as the distribution of $\eta(X)$.

Consider a fixed partition $\omega_0 = 0 < \omega_1 \le \ldots \le \omega_K \le 1 = \omega_{K+1}$ of the interval $(0,1)$. We can then construct an estimator of $s^*$ by overlaying a finite collection of (estimated) density level sets $\hat{C}_{\omega_1}, \ldots, \hat{C}_{\omega_K}$:

$$\hat{s}(x) = \sum_{i=1}^K \mathbb{I}\{x \in \hat{C}_{\omega_i}\},$$

which may be seen as an empirical version of a discrete version of the target $s^*$.

In order to consider the performance of such an estimator, we need to compare the ROC curve of $\hat{s}$ to the optimal ROC curve. However, if the sequence $\{\hat{C}_{\omega_i}\}_{i=1,\ldots,K}$ is not decreasing, the computation of the ROC curve as a function of the errors of the overlaying classifiers becomes complicated.

The main result of the paper is the next theorem which is proved for a modified sequence which yields to a different estimator. We introduce: $\{\tilde{C}_{\omega_i}\}_{1 \le i \le K}$ defined by:

$$\tilde{C}_{\omega_1} = \hat{C}_{\omega_1} \ \text{ and } \ \tilde{C}_{\omega_{i+1}} = \tilde{C}_{\omega_i} \cup \hat{C}_{\omega_{i+1}} \ \text{ for all } i \in \{1,\ldots,K-1\}\,.$$

The corresponding scoring function is then given by:

$$\tilde{s}_K(x) = \sum_{i=1}^K \mathbb{I}\{x \in \tilde{C}_{\omega_i}\}\,. \tag{2}$$

Hence, the ROC curve of $\tilde{s}_K$ is simply the broken line that connects the knots $(\alpha(\tilde{C}_{\omega_i}), \beta(\tilde{C}_{\omega_i}))$, $0 \leq i \leq K + 1$.

The next result offers a rate bound in the ROC space, equipped with a sup-norm. Up to our knowledge, this is the first result on the generalization ability of decision rules in such a functional space.

**Theorem 2** *Under the same assumptions as in Theorem 1 and with the previous notations, we set* $K = K_n \sim n^{1/8}$. *Fix* $\epsilon > 0$. *Then, there exists a constant $c$ such that, with probability at least* $1 - \delta$, *we have:*

$$\sup_{\alpha \in [\epsilon, 1-\epsilon]} |\mathrm{ROC}^*(\alpha) - \mathrm{ROC}(\tilde{s}_K, \alpha)| \leq \frac{c \log(1/\delta)}{\epsilon n^{1/4}} .$$

**Remark 1** (PERFORMANCE OF CLASSIFIERS AND ROC CURVES.) In the present paper, we have adopted a scoring approach to ROC analysis which is somehow related to the evaluation of the performance of classifiers in ROC space. Using combinations of such classifiers to improve performance in terms of ROC curves has also been pointed out in [BDH06] and [BCT07].

**Remark 2** (PLUG-IN ESTIMATOR OF THE REGRESSION FUNCTION.) Note that taking $\nu = \lambda$ the Lebesgue measure over $[0,1]$ in the expression of $s^*$ leads to the regression function $\eta(x) = \int \mathbb{I}\{x \in C^*_\omega\} \, d\omega$. An estimator for the regression function could be the following: $\hat{\eta}_K(x) = \sum_{i=1}^{K+1} (\omega_i - \omega_{i-1}) \mathbb{I}\{x \in \tilde{C}_{\omega_i}\}$.

**Remark 3** (ADAPTIVITY OF THE PARTITION.) A natural extension of the approach would be to consider a flexible partition $(\omega_i)_i$ which could possibly be adaptively chosen depending on the local regularity of the ROC curve. For now, it is not clear how to extend the method of the paper to take into account adaptive partitions, however we have investigated such partitions corresponding to different approximation schemes of the optimal ROC curve elsewhere ([CV08]), but the rates of convergence obtained in the present paper are faster.

**Optimal ROC curve approximation and estimation.** We now provide some insights on the previous result. The key for the proof of Theorem 2 is the idea of a piecewise linear approximation of the optimal ROC curve.

We introduce some notations. Let $\omega_0 = 0 < \omega_1 < \ldots < \omega_K < \omega_{K+1} = 1$ be a given partition of $[0,1]$ such that $\max_{i \in \{0, \ldots, K\}} \{\omega_{i+1} - \omega_i\} \leq \delta$. Set: $\forall i \in \{0, \ldots, K+1\}$, $\alpha_i^* = \alpha(C^*_{\omega_i})$ and $\beta_i^* = \beta(C^*_{\omega_i})$.

The broken line that connects the knots $\{(\alpha_i^*, \beta_i^*); \ 0 \leq i \leq K+1\}$ provides a piecewise linear (concave) approximation/interpolation of the optimal ROC curve $\mathrm{ROC}^*$. In the spirit of the finite element method (FEM, see [dB01] for instance), we introduce the "hat functions" defined by:

$$\forall i \in \{1, \ldots, K-1\}, \quad \phi_i^*(\cdot) = \phi(\cdot; (\alpha_{i-1}^*, \alpha_i^*)) - \phi(\cdot; (\alpha_i^*, \alpha_{i+1}^*)),$$

with the notation $\phi(\alpha, (\alpha_1, \alpha_2)) = (\alpha - \alpha_1)/(\alpha_2 - \alpha_1) \cdot \mathbb{I}\{\alpha \in [\alpha_1, \alpha_2]\}$ for all $\alpha_1 < \alpha_2$. We also set $\phi_K^*(\cdot) = \phi(\cdot; (\alpha_K^*, 1))$ for notational convenience. The piecewise linear approximation of $\mathrm{ROC}^*$ may then be written as:

$$\widetilde{\mathrm{ROC}}^*(\alpha) = \sum_{i=1}^{K} \beta_i^* \phi_i^*(\alpha) .$$

In order to obtain an empirical estimator of $\widetilde{\mathrm{ROC}}^*(\alpha)$, we propose: i) to find an estimate $\hat{C}_{\omega_i}$ of the true level set $C^*_{\omega_i}$ based on the training sample $\{(X_i, Y_i)\}_{i=1,\ldots,n}$ as in Section 3, ii) to compute the corresponding errors $\hat{\alpha}_i$ and $\hat{\beta}_i$ using a test sample $\{(X_i', Y_i')\}_{i=1,\ldots,n}$. Hence we define:

$$\hat{\alpha}_i(C) = \frac{1}{n_-} \sum_{i=1}^{n} \mathbb{I}\{X_i' \in C, Y_i' = -1\} \ \text{and} \ \hat{\beta}_i(C) = \frac{1}{n_+} \sum_{i=1}^{n} \mathbb{I}\{X_i' \in C, Y_i' = +1\},$$

with $n_+ = \sum_{i=1}^{n} \mathbb{I}\{Y_i' = +1\} = n - n_-$. We set $\hat{\alpha}_i = \hat{\alpha}_i(\hat{C}_{\omega_i})$ and $\hat{\beta}_i = \hat{\beta}_i(\hat{C}_{\omega_i})$. We propose the following estimator of $\widetilde{\mathrm{ROC}}^*(\alpha)$:

$$\widehat{\mathrm{ROC}^*}(\alpha) = \sum_{i=1}^{K} \hat{\beta}_i \hat{\phi}_i(\alpha),$$

where $\hat{\phi}_K(\alpha) = \phi(.; (\hat{\alpha}_K, 1))$ and $\hat{\phi}_i(\alpha) = \phi(.; (\hat{\alpha}_{i-1}, \hat{\alpha}_i)) - \phi(.; (\hat{\alpha}_i, \hat{\alpha}_{i+1}))$ for $1 \leq i < K$. Hence, $\widehat{\mathrm{ROC}}$ is the broken line connecting the empirical knots $\{(\hat{\alpha}_i, \hat{\beta}_i); \ 0 \leq i \leq K+1\}$.

The next result takes the form of a deviation bound for the estimation of the optimal ROC curve. It quantifies the order of magnitude of a confidence band in supremum norm around an empirical estimate based on the previous approximation scheme with empirical counterparts.

**Theorem 3** *Under the same assumptions as in Theorem 1 and with the previous notations, set $K = K_n \sim n^{1/6}$. Fix $\epsilon > 0$. Then, there exists a constant $c$ such that, with probability at least $1 - \delta$,*

$$\sup_{\alpha \in [\epsilon, 1-\epsilon]} |\widehat{\mathrm{ROC}^*}(\alpha) - \mathrm{ROC}^*(\alpha)| \leq c\epsilon^{-1} \left( \frac{\log(\mathrm{n}/\delta)}{\mathrm{n}} \right)^{1/3} .$$

## 5   Conclusion

We have provided a strategy based on overlaid classifiers to build a nearly-optimal scoring function. Statistical guarantees are provided in terms of rates of convergence for a functional criterion which is the ROC space equipped with a supremum norm. This is the first theoretical result of this nature. To conclude, we point out that ROC analysis raises important and novel issues for statistical learning and we hope that the present contribution gives a flavor of possible research directions.

## Appendix - Proof section

**Proof of Theorem 1.** The idea of the proof is to relate the excess risk in terms of $\alpha$-error to the excess risk in terms of weighted classification error. First we re-parameterize the weighted classification error. Set $C(\alpha) = \{x \in \mathcal{X} \mid \eta(x) > Q^*(\alpha)\}$ and:

$$\ell_\omega(\alpha) = L_\omega(C(\alpha)) = 2(1-p)\omega \, \alpha + 2p(1-\omega)(1 - \mathrm{ROC}^*(\alpha))$$

Since $\mathrm{ROC}^*$ is assumed to be differentiable and using Proposition 3, it is easy to check that the value $\alpha^* = \alpha(C_\omega^*)$ minimizes $\ell_\omega(\alpha)$. Denote by $\ell_\omega^* = \ell_\omega(\alpha^*)$. It follows from a Taylor expansion of $\ell_\omega(\alpha)$ around $\alpha^*$ at the second order that there exists $\alpha_0 \in [0, 1]$ such that:

$$\ell_\omega(\alpha) = \ell_\omega^* - p(1-\omega) \, \frac{d^2}{d\alpha^2} \mathrm{ROC}^*(\alpha_0) \, (\alpha - \alpha^*)^2$$

Using also the fact that $\mathrm{ROC}^*$ dominates any other curve of the ROC space, we have: $\forall C \subset \mathcal{X}$ measurable, $\beta(C) \leq \mathrm{ROC}^*(\alpha(\mathrm{C}))$. Also, by assumption, there exists $m$ such that: $\forall \alpha \in [0, 1]$, $\frac{d^2}{d\alpha^2} \mathrm{ROC}^*(\alpha) \geq -\mathrm{m}$. Hence, since $\ell_\omega(\alpha(\hat{C}_\omega)) = L_\omega(\hat{C}_\omega)$, we have:

$$\left( \alpha(\hat{C}_\omega) - \alpha(C_\omega^*) \right)^2 \leq \frac{1}{mp(1-\omega)} \left( L_\omega(\hat{C}_\omega) - L_\omega(C_\omega^*) \right) .$$

We have obtained the desired inequality. It remains to get the rate of convergence for the weighted empirical risk.

Now set: $F^* = pG^* + (1-p)H^*$. We observe that: $\forall t > 0$, $\mathbb{P}(|\eta(X) - \omega| \leq t) = F^*(\omega + t) - F^*(\omega - t) \leq 2t \, \sup_u (F^*)'(u)$. We have thus shown that the distribution satisfies a modified Tsybakov's margin condition [Tsy04], for all $\omega \in [0, 1]$, of the form:

$$\mathbb{P}(|\eta(X) - \omega| \leq t) \leq D \, t^{\frac{\gamma}{1-\gamma}}.$$

with $\gamma = 1/2$ and $D = 2\sup_u (F^*)'(u)$. Adapting slightly the argument used in [Tsy04], [BBL05], we have that, under the modified margin condition, there exists a constant $c$ such that, with probability $1 - \delta$:

$$L_\omega(\hat{C}_\omega) - L_\omega^*(C_\omega^*) \leq c \left( \frac{\log(1/\delta)}{n} \right)^{\frac{1}{2-\gamma}} .$$

**Proof of Theorem 2.** We note $\tilde{\alpha}_i = \alpha(\tilde{C}_{\omega_i})$, $\tilde{\beta}_i = \beta(\tilde{C}_{\omega_i})$ and also $\tilde{\phi}_i(\,\cdot\,) = \phi(\,\cdot\,; (\tilde{\alpha}_{i-1}, \tilde{\alpha}_i)) - \phi(\,\cdot\,; (\tilde{\alpha}_i, \tilde{\alpha}_{i+1}))$. We then have $\mathrm{ROC}(\tilde{s}_K, \alpha) = \sum_{i=1}^K \tilde{\beta}_i \tilde{\phi}_i(\alpha)$ and we can use the following

decomposition, for any $\alpha \in [0,1]$:

$$\mathrm{ROC}^*(\alpha) - \mathrm{ROC}(\tilde{s}_K, \alpha) = \left( \mathrm{ROC}^*(\alpha) - \sum_{i=1}^{K} \mathrm{ROC}^*(\tilde{\alpha}_i)\tilde{\phi}_i(\alpha) \right) + \sum_{i=1}^{K} (\mathrm{ROC}^*(\tilde{\alpha}_i) - \tilde{\beta}_i)\tilde{\phi}_i(\alpha) .$$

It is well-known folklore in linear approximation theory ([dB01]) that if $\tilde{s}_K$ is a piecewise constant scoring function whose ROC curve interpolates the points $\{(\tilde{\alpha}_i, \mathrm{ROC}^*(\tilde{\alpha}_i))\}_{i=0,\dots,K}$ of the optimal ROC curve, then we can bound the first term (which is positive), $\forall \alpha \in [0,1]$, by:

$$-\frac{1}{8} \inf_{\alpha \in [0,1]} \frac{d^2}{d\alpha^2} \mathrm{ROC}^*(\alpha) \cdot \max_{0 \le i \le K} (\tilde{\alpha}_{i+1} - \tilde{\alpha}_i)^2 .$$

Now, to control the second term, we upper bound the following quantity:

$$|\mathrm{ROC}^*(\tilde{\alpha}_i) - \tilde{\beta}_i| \le \sup_{\alpha \in [0,1]} \frac{d}{d\alpha} \mathrm{ROC}^*(\alpha) \cdot |\tilde{\alpha}_i - \alpha_i^*| + |\beta_i^* - \tilde{\beta}_i|$$

We further bound: $|\tilde{\alpha}_i - \alpha_i^*| \le |\tilde{\alpha}_i - \alpha_i| + |\alpha_i - \alpha_i^*|$ where $\alpha_i = \alpha(\hat{C}_i)$. In order to deal with the first term, the next lemma will be needed:

**Lemma 1** *We have, for all $k \in \{1, \dots, K\}$:*

$$\alpha(\tilde{C}_k) = \alpha(\hat{C}_k) + (k-1)O_{\mathbb{P}}(n^{-1/4}) .$$

*where the notation $O_{\mathbb{P}}(1)$ is used for a r.v. which is bounded in probability.*

From the lemma, it follows that: $\max_{1 \le i \le K} |\tilde{\alpha}_i - \alpha_i| = O_{\mathbb{P}}(Kn^{-1/4})$. We can then use Theorem 1 with $\delta$ replaced by $\delta/K$ to get that $\max_{1 \le i \le K} |\alpha_i - \alpha_i^*| = O_{\mathbb{P}}((n^{-1}\log K)^{1/3})$. The same inequalities hold with the $\beta$'s. It remains to control the quantity $\tilde{\alpha}_{i+1} - \tilde{\alpha}_i$. We have:

$$| \tilde{\alpha}_{i+1} - \tilde{\alpha}_i | \le \max_{1 \le k \le K} | \alpha(\hat{C}_k) - \alpha(\hat{C}_{k-1}) | + K\, O_{\mathbb{P}}(n^{-1/4}) .$$

We have that:

$$\max_{1 \le k \le K} | \alpha(\hat{C}_k) - \alpha(\hat{C}_{k-1}) | \le 2 \max_{1 \le k \le K} | \alpha(\hat{C}_k) - \alpha(C_k^*) | + \max_{1 \le k \le K} | \alpha(C_k^*) - \alpha(C_{k-1}^*) |$$

As before, we have that the first term is of the order $(\log K/n)^{1/3}$ and since the second derivative of the optimal ROC curve is bounded, the second term is of the order $K^{-1}$. Eventually, we choose $K$ in order to optimize the quantity: $K^{-2} + (\log K/n)^{2/3} + K^2 n^{-1/2} + Kn^{-1/4} + (\log K/n)^{1/3}$. As only the first and the third term matter, this leads to the choice of $K = K_n \sim n^{1/8}$.

**Proof of Lemma 1.**

We have that $\alpha(\tilde{C}_2) = \alpha(\hat{C}_2) + \alpha(\hat{C}_1 \setminus \hat{C}_2)$. Therefore, since $C_1^* \subset C_2^*$ and observing that

$$\alpha(\hat{C}_1 \setminus \hat{C}_2) = \alpha(((\hat{C}_1 \setminus C_1^*) \cup (\hat{C}_1 \cap C_1^*)) \setminus ((\hat{C}_2 \setminus C_2^*) \cup (\hat{C}_2 \cap C_2^*))) ,$$

it suffices to use the additivity of the probability measure $\alpha(.)$ to get: $\alpha(\tilde{C}_2) = \alpha(\hat{C}_2) + O_{\mathbb{P}}(n^{-1/4})$. Eventually, errors are stacked and we obtain the result.

**Proof of Theorem 3.**

We use the following decomposition, for any fixed $\alpha \in (0,1)$:

$$\widehat{\mathrm{ROC}^*}(\alpha) - \mathrm{ROC}^*(\alpha) = \left( \widehat{\mathrm{ROC}^*}(\alpha) - \sum_{i=1}^{K} \mathrm{ROC}^*(\hat{\alpha}_i)\hat{\phi}_i(\alpha) \right) + \left( \sum_{i=1}^{K} \mathrm{ROC}^*(\hat{\alpha}_i)\hat{\phi}_i(\alpha) - \mathrm{ROC}^*(\alpha) \right) .$$

Therefore, we have by a triangular inequality: $\forall \alpha \in [0,1]$,

$$\left| \widehat{\mathrm{ROC}^*}(\alpha) - \sum_{i=1}^{K} \mathrm{ROC}^*(\hat{\alpha}_i)\hat{\phi}_i(\alpha) \right| \le \max_{1 \le i \le K} |\hat{\beta}_i - \beta_i| + |\beta_i - \beta_i^*| + |\mathrm{ROC}^*(\alpha_i^*) - \mathrm{ROC}^*(\hat{\alpha}_i)| .$$

And, by the finite increments theorem, we have:

$$|\text{ROC}^*(\alpha_i^*) - \text{ROC}^*(\hat{\alpha}_i)| \leq \left( \sup_{\alpha \in [0,1]} \frac{\mathrm{d}}{\mathrm{d}\alpha} \text{ROC}^*(\alpha) \right) \; (|\alpha_i^* - \alpha_i| + |\alpha_i - \hat{\alpha}_i|) \; .$$

For the other term, we use the same result on approximation as in the proof of Theorem 2:

$$\left| \sum_{i=1}^{K} \text{ROC}^*(\hat{\alpha}_i)\hat{\phi}_i(\alpha) - \text{ROC}^*(\alpha) \right| \leq -\frac{1}{8} \inf_{\alpha \in [0,1]} \frac{d^2}{d\alpha^2} \text{ROC}^*(\alpha) \cdot \max_{0 \leq i \leq K} (\hat{\alpha}_{i+1} - \hat{\alpha}_i)^2$$

$$\max_{0 \leq i \leq K} (\hat{\alpha}_{i+1} - \hat{\alpha}_i) \leq \max_{0 \leq i \leq K} (\alpha_{i+1}^* - \alpha_i^*) + 2 \max_{1 \leq i \leq K} |\alpha_i^* - \alpha_i| + 2 \max_{1 \leq i \leq K} |\hat{\alpha}_i - \alpha_i| \; .$$

We recall that: $\max_{1 \leq i \leq K} |\hat{\alpha}_i - \alpha_i|. = O_{\mathbb{P}}(Kn^{-1/2})$. Moreover, $\max_{0 \leq i \leq K} \{\alpha_{i+1}^* - \alpha_i^*\}$ is of the order of $K^{-1}$. And with probability at least $1 - \delta$, we have that $\max_{1 \leq i \leq K} |\alpha_i^* - \alpha_i|$ is bounded as in Theorem 1, except that $\delta$ is replaced by $\delta/K$ in the bound. Eventually, we get the generalization bound: $K^{-2} + (\log K/n)^{1/3}$, which is optimal for a number of knots: $K \sim n^{1/6}$.

## References

[AA07]      J.-Y. Audibert and A.Tsybakov. Fast learning rates for plug-in classifiers. *Annals of statistics*, 35(2):608–633, 2007.

[AGH+05]  S. Agarwal, T. Graepel, R. Herbrich, S. Har-Peled, and D. Roth. Generalization bounds for the area under the ROC curve. *J. Mach. Learn. Res.*, 6:393–425, 2005.

[BBL05]     S. Boucheron, O. Bousquet, and G. Lugosi. Theory of Classification: A Survey of Some Recent Advances. *ESAIM: Probability and Statistics*, 9:323–375, 2005.

[BCT07]     M. Barreno, A.A. Cardenas, and J.D. Tygar. Optimal ROC curve for a combination of classifiers. In *NIPS'07*, 2007.

[BDH06]     F.R. Bach, D.Heckerman, and Eric Horvitz. Considering cost asymmetry in learning classifiers. *Journal of Machine Learning Research*, 7:1713–1741, 2006.

[Cav97]     L. Cavalier. Nonparametric estimation of regression level sets. *Statistics*, 29:131–160, 1997.

[CLV08]     S. Clémençon, G. Lugosi, and N. Vayatis. Ranking and empirical risk minimization of U-statistics. *The Annals of Statistics*, 36(2):844–874, 2008.

[CV07]      S. Clémençon and N. Vayatis. Ranking the best instances. *Journal of Machine Learning Research*, 8:2671–2699, 2007.

[CV08]      S. Clémençon and N. Vayatis. Tree-structured ranking rules and approximation of the optimal ROC curve. Technical Report hal-00268068, HAL, 2008.

[dB01]      C. de Boor. *A practical guide to splines*. Springer, 2001.

[Ega75]     J.P. Egan. *Signal Detection Theory and ROC Analysis*. Academic Press, 1975.

[FISS03]    Y. Freund, R. D. Iyer, R. E. Schapire, and Y. Singer. An efficient boosting algorithm for combining preferences. *Journal of Machine Learning Research*, 4:933–969, 2003.

[RV06]      P. Rigollet and R. Vert. Fast rates for plug-in estimators of density level sets. Technical Report arXiv:math/0611473v2, arXiv:math/0611473v2, 2006.

[Tsy04]     A. Tsybakov. Optimal aggregation of classifiers in statistical learning. *Annals of Statistics*, 32(1):135–166, 2004.

[vT68]      H.L. van Trees. *Detection, Estimation, and Modulation Theory, Part I*. Wiley, 1968.

[WN07]      R. Willett and R. Nowak. Minimax optimal level set estimation. *IEEE Transactions on Image Processing*, 16(12):2965–2979, 2007.

